# A Framework for the Cooperation of Learning Algorithms

Léon Bottou

Patrick Gallinari

Laboratoire de Recherche en Informatique
Université de Paris XI
91405 Orsay Cedex
France

## Abstract

We introduce a framework for training architectures composed of several modules. This framework, which uses a statistical formulation of learning systems, provides a unique formalism for describing many classical connectionist algorithms as well as complex systems where several algorithms interact. It allows to design hybrid systems which combine the advantages of connectionist algorithms as well as other learning algorithms.

## 1 INTRODUCTION

Many recent achievements in the connectionist area have been carried out by designing systems where different algorithms interact. For example (Bourlard & Morgan, 1991) have mixed a Multi-Layer Perceptron (MLP) with a Dynamic Programming algorithm. Another impressive application (Le Cun, Boser & al., 1990) uses a very complex multi-layer architecture, followed by some statistical decision process. Also, in speech or image recognition systems, input signals are sequentially processed through different modules. Modular systems are the most promising way to achieve such complex tasks. They can be built using simple components and therefore can be easily modified or extended, also they allow to incorporate into their architecture some *structural a priori knowledge* about the task decomposition. Of course, this is also true for connectionism, and important

progresses in this field could be achieved if we were able to train multi-modules architectures.

In this paper, we introduce a formal framework for designing and training such cooperative systems. It provides a unique formalism for describing both the different modules and the global system. We show that it is suitable for many connectionist algorithms, which allows to make them cooperate in an optimal way according to the goal of learning. It also allows to train hybrid systems where connectionist and classical algorithms interact. Our formulation is based on a probabilistic approach to the problem of learning which is described in section 2. One of the advantages of this approach is to provide a formal definition of the *goal of learning*. In section 3, we introduce modular architectures where each module can be described using this framework, and we derive explicit formulas for training the global system through a stochastic gradient descent algorithm. Section 4 is devoted to examples, including the case of hybrid algorithms combining MLP and Learning Vector Quantization (Bollivier, Gallinari & Thiria, 1990).

## 2   LEARNING SYSTEMS

The probabilistic formulation of the problem of learning has been extensively studied for three decades (Tsypkin 1971), and applied to control, pattern recognition and adaptive signal processing. We recall here the main ideas and refer to (Tsypkin 1971) for a detailed presentation.

### 2.1   EXPECTED COST

Let **x** be an instance of the concept to learn. In the case of a pattern recognition problem for example, **x** would be a pair (pattern, class). The concept is mathematically defined by an unknown probability density function $p(\mathbf{x})$ which measures the likelihood of instance **x**.

We shall use a system parameterized by **w** to perform some task that depends on $p(\mathbf{x})$. Given an example **x**, we can define a *local cost*, $J(\mathbf{x},\mathbf{w})$, that measures how well our system behaves on that example. For instance, for classification J would be zero if the system puts a pattern in the correct class, or positive in case of misclassification.

Learning consists in finding a parameter $\mathbf{w}^*$ that optimises some functional of the model parameters. For instance, one would like to minimize the *expected cost* (1).

$$C(\mathbf{w}) = \int J(\mathbf{x},\mathbf{w})\, p(\mathbf{x}) d\mathbf{x} \tag{1}$$

The expected cost cannot be explicitly computed, because the density $p(\mathbf{x})$ is unknown. Our only knowledge of the process comes from a series of observations $\{\mathbf{x}_1 \ldots \mathbf{x}_n\}$ drawn from the unknown density $p(\mathbf{x})$. Therefore, the quality of our system can only be measured through the realisations $J(\mathbf{x},\mathbf{w})$ of the local cost function for the different observations.

## 2.2   STOCHASTIC GRADIENT DESCENT

Gradient descent algorithms are the simplest minimization algorithms. We cannot, however, compute the gradient of the expected cost (1), because $p(\mathbf{x})$ is unknown. Estimating these derivatives on a training set $\{\mathbf{x}_1...\mathbf{x}_n\}$, gives the gradient algorithm (2), where $\nabla J$ denotes the gradient of $J(\mathbf{x},\mathbf{w})$ with respect to $\mathbf{w}$, and $\varepsilon_t$, a small positive constant, the "learning rate".

$$w_{t+1} = w_t - \varepsilon_t \frac{1}{n} \sum_{i=1}^{n} \nabla J(\mathbf{x}_i, \mathbf{w}_t) \tag{2}$$

The *stochastic gradient descent* algorithm (3) is an alternative to algorithm (2). At each iteration, an example $\mathbf{x}_t$ is drawn at random, and a new value of $\mathbf{w}$ is computed.

$$w_{t+1} = w_t - \varepsilon_t \nabla J(\mathbf{x}_t, \mathbf{w}_t) \tag{3}$$

Algorithm (3) is faster and more reliable than (2), it is the only solution for training adaptive systems like Neural networks (NN). Such stochastic approximations have been extensively studied in adpative signal processing (Benveniste, Metiver & Priouret, 1987), (Ljung & Soderström, 1983). Under certain conditions, algorithm (3) converges almost surely (Bottou, 1991), (White, 1991) and allows to reach an optimal state of the system.

# 3   MODULAR LEARNING SYSTEMS

Most often, when the goal of learning is complex, it can be achieved more easily by using a decomposition of the global task into several simpler subtasks which for instance reflect some a priori knowledge about the structure of the problem. One can use this decomposition to build modular architectures where each module will correspond to one of the subtasks.

Within this framework, we will use the expected risk (1) as the goal of learning. The problem now is to change the analytical formulation of the functional (1) so as to introduce the modular decomposition of the global task. In (1), the analytic expression of the local cost $J(\mathbf{x},\mathbf{w})$ has two meanings: it describes a parametric relationship between the inputs and the outputs of the system, and measures the quality of the system. To introduce the decomposition, one may write this local cost $J(\mathbf{x},\mathbf{w})$ as the composition of several functions. One of them will take into account the local error and therefore measure the quality of the system; the others will correspond to the decomposition of the parametric relationship between the inputs and the outputs of the system (Figure 1). Each of the modules will therefore receive some inputs from other modules or the external world and produce some outputs which will be sent to other modules.

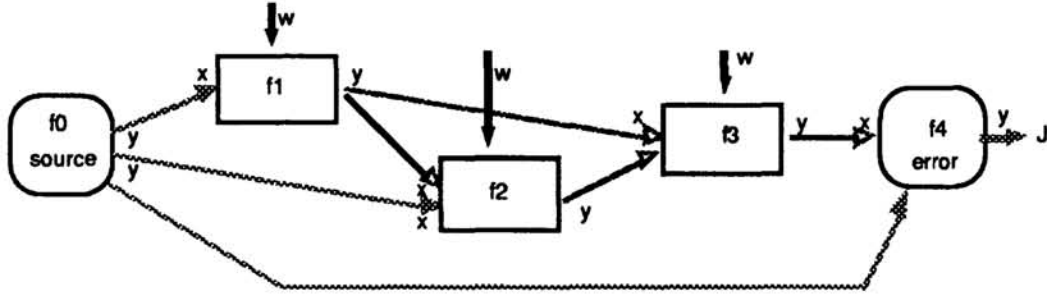

Figure 1: A modular system

In classical systems, these modules correspond to well defined processing stages like e.g. signal processing, filtering, feature extraction, classification. They are trained sequentially and then linked together to build a complete processing system which takes some inputs (e.g. raw signals) and produces some outputs (e.g. classes). Neither the assumed decomposition, nor the behavior of the different modules is guaranteed to optimally contribute to the global goal of learning. We will show in the following that it is possible to optimally train such systems.

## 3.1 TRAINING MODULAR SYSTEMS

Each function in the above composition defines a local processing stage or module whose outputs are defined by a parametric function of its inputs (4).

$$\forall j \in Y^{-1}(n), \ y_j = f_j \Big( (x_k) \, _{k \in X^{-1}(n)} \, , \, (w_i) \, _{i \in W^{-1}(n)} \Big) \tag{4}$$

$Y^{-1}(n)$ ( resp. $X^{-1}(n)$, and $W^{-1}(n)$ ) denotes the set of subscripts associated to the outputs $y$ ( resp. inputs $x$ and parameters $w$ ) of module $n$. Conversely, output $y_j$ ( resp. input $x_k$ and parameter $w_i$ ) belongs to module $Y(j)$ ( resp. $X(k)$ and $W(i)$ ).

Modules are linked so as to build a feed-forward topology which is expressed by a function $\phi$.

$$\forall k, \ x_k = y_{\phi(k)} \tag{5}$$

We shall consider that the first module only feeds the system with examples and that the last module only computes $y_{last} = J(x, w)$.

Following (Le Cun, 1988), we can compute the derivatives of $J$ with a Lagrangian method. Let $\alpha$ and $\beta$ be the Lagrange coefficients for constraints (4) and (5).

$$L = J - \sum_k \beta_k(x_k - y_{\phi(k)}) - \sum_j \alpha_j \Big( y_j - f_j \Big( (x_k) \, _{k \in X^{-1}Y(j)}, \, (w_i) \, _{i \in W^{-1}Y(j)} \Big) \Big) \tag{6}$$

By equating the derivatives of $L$ with respect to $x$ and $y$ to zero, we get recursive formulas for computing $\alpha$ and $\beta$ in a single backward pass along the acyclic graph $\phi$.

$$\alpha_{last} = 1, \qquad \beta_k = \sum_{j \in Y^{-1}X(k)} \alpha_j \frac{\partial f_j}{\partial x_k}, \qquad \alpha_j = \sum_{k \in \phi^{-1}(j)} \beta_k \quad (\text{if } j \neq last) \tag{7}$$

Then, the derivatives of J with respect to the weights are:

$$\frac{\partial J}{\partial w_i}(w) = \frac{\partial L}{\partial w_i}(\alpha, \beta, w) = \sum_{j \in Y^{-1}W(i)} \alpha_j \frac{\partial f_j}{\partial w_i} \tag{8}$$

Once we have computed the derivatives of the local cost $J(\mathbf{x}, \mathbf{w})$, we can apply the stochastic gradient descent algorithm (3) for minimizing of the expected cost $C(\mathbf{w})$.

We shall say that each module is defined by the equations in (7) and (8) that characterize its behavior. These equations are:

- a forward equation (F)

$$y_j = f_j(\,(x_k)_{k \in X^{-1}(n)}\,,\, (w_i)_{i \in W^{-1}(n)}\,)$$

- a backward equation (B)

$$\beta_k = \sum_{j \in Y^{-1}X(k)} \alpha_j \frac{\partial f_j}{\partial x_k}$$

- a gradient equation (G)

$$\Delta_i = \frac{\partial J}{\partial w_i} = \sum_{j \in Y^{-1}W(i)} \alpha_j \frac{\partial f_j}{\partial w_i}$$

The remaining equations do not depend on the nature of the modules. They describe how modules interact during training. Like back-propagation, they address the credit assignment problem between modules by globally minimizing a single cost function. Training such a complex system actually consists in *cooperatively training* its components.

## 4   EXAMPLES

Most learning algorithms, as well as new algorithms may be expressed as modular learning systems. Here are some simple examples of modules and systems.

### 4.1   LINEAR AND QUASI-LINEAR SYSTEMS

| MODULE | SYMBOL | FORWARD | BACKWARD | GRADIENT |
|---|---|---|---|---|
| Matrix product | Wx | $y_i = \sum_k w_{ik} x_k$ | $\beta_k = \sum_i \alpha_i w_{ik}$ | $\Delta_{ik} = \alpha_i x_k$ |
| Mean square error | MSE | $J = \sum_k (d_k - x_k)^2$ | $\beta_k = -2(d_k - x_k)$ | --- |
| Perceptron error | Perceptron | $J = -\sum_k (d_k - 1_{\Re^+}(x_k)) x_k$ | $\beta_k = -(d_k - 1_{\Re^+}(x_k))$ | --- |
| Sigmoid | sigmoïd | $y_k = f(x_k)$ | $\beta_k = f'(x_k) \alpha_k$ | --- |

A few basic modules are defined in the above table. Figure 2 gives examples of linear and quasi linear algorithms derived by combining these modules.

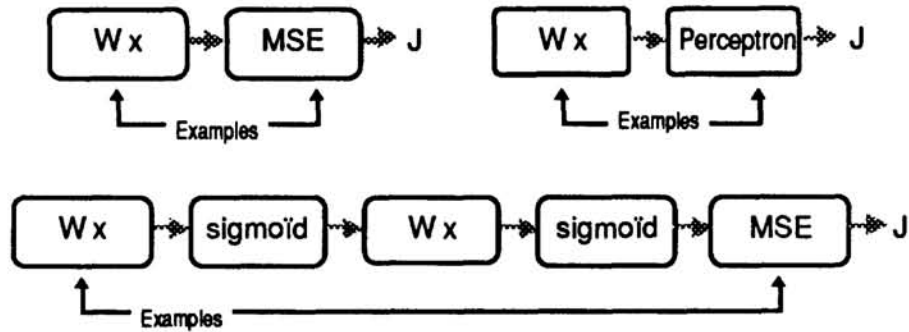

Figure 2: An Adaline, a Perceptron, and a 2-Layer Perceptron.

Some MLP architectures, Time Delay Networks for instance, use local connections and shared weights. Such complex architectures may be constructed by defining either quasi-linear unit modules or complex matrix operations modules like convolutions. The latter solution leads to more efficient implementations. Figure 3 gives an example of convolution module, composed of several matrix products modules.

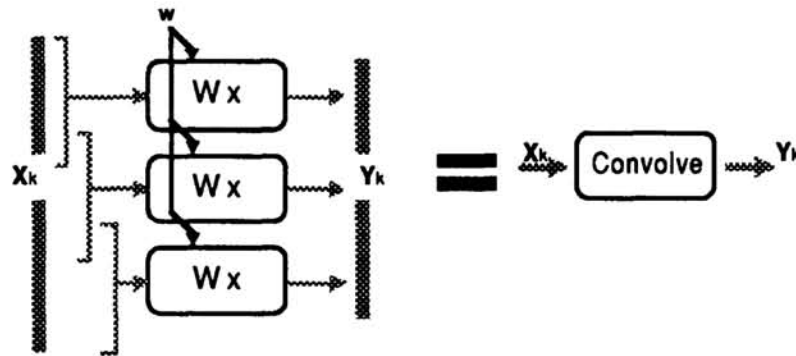

Figure 3: A convolution module, composed of several matrix product modules.

## 4.2 EUCLIDIAN DISTANCE BASED ALGORITHMS

A wide class of learning systems are based on the measure of euclidian distances. Again, defining an euclidian distance module and some adequate cost functions allows for handling most euclidian distance based algorithms. Here are some examples:

| MODULE | SYMBOL | FORWARD | BACKWARD | GRADIENT |
|---|---|---|---|---|
| Euclidian distance | $(x-w)^2$ | $y_j = \sum_k (w_{jk} x_k)^2$ | $\beta_k = -2\sum_k \alpha_j (w_{jk} - x_k)$ | $\Delta_{jk} = 2\alpha_j (w_{jk} - x_k)$ |
| Minimum | Min | $J = x_{k^*} = \text{Min}\{x_k\}$ | $\beta_{k^*} = 1, \beta_{k \neq k^*} = 0$ | --- |
| LVQ 1 error | LVQ1 | If the nearest reference $x_{k^*}$ is associated to the correct class | | |
| | | $J = x_{k^*} = \text{Min}\{x_k\}$ | $\beta_{k^*} = 1, \beta_{k \neq k^*} = 0$ | --- |
| | | else | | |
| | | $J = -x_{k^*} = -\text{Min}\{x_k\}$ | $\beta_{k^*} = -1, \beta_{k \neq k^*} = 0$ | --- |

Combining an euclidian distance module with a "minimum" error module gives a K-means algorithm; combining it with a LVQ1 error module gives the LVQ1 algorithm (Figure 4).

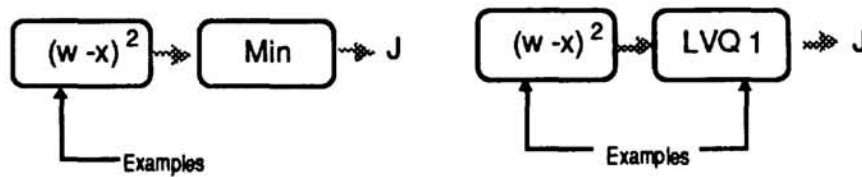

Figure 4: K-Means and Learning Vector Quantization.

## 4.3  HYBRID ALGORITHMS

Hybrid algorithms which may combine classical and connectionist learning algorithms are easily defined by chaining appropriate modules. Figure 5, for instance, depicts an algorithm combining a MLP layer and LVQ1. This algorithm has been described and empirically compared to other pattern recognition algorithms in (Bollivier, Gallinari & Thiria, 1990).

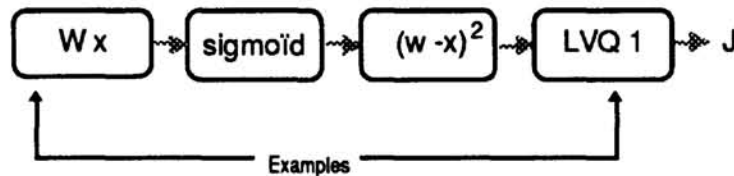

Figure 5: An hybrid algorithm combining a MLP and LVQ.

Cooperative training gives a framework and a possible implementation for such algorithms. Nevertheless, there are still specific problems (e.g. convergence, initialization) which require a careful study. More complex hybrid systems, including combinations of Markov Models and Time Delay Networks, have been described within this framework in (Bottou,1991).

## 5  CONCLUSION

Cooperative training of modular systems provides a unified view of many learning algorithms, as well as hybrid systems which combine classical or connectionist algorithms. Our formalism provides a way to define specific modules and to combine them into a cooperative system. This allows to design and implement complex learning systems which eventually incorporate structural a priori knowledge about the task.

### Acknowledgements

During this work, L.B. was supported by DRET grant n° 87/808/19.

### References

Benveniste A., Metivier M., Priouret P. (1987) *Algorithmes adaptatifs et approximations stochastiques*, Masson

Bollivier M. de, Gallinari P. & Thiria S. (1990) Cooperation of neural nets for robust classification, *Procedings of IJCNN 90*, San Diego, vol1, 113-120.

Bottou L. (1991) *Une approche théorique de l'apprentissage connexionniste; applications à la reconnaissance de la parole.* PhD Thesis, Université de Paris XI

Bourlard H., Morgan N. (1991) A Continuous Speech Recognition System Embedding MLP into HMM - In Touretzky D.S., Lipmann R. (eds.) *Advances in Neural Information Processing Systems 3* (this volume), Morgan-Kaufman

Le Cun Y.: A theoretical framework for back-propagation (1988) in Touretzky D., Hinton G. & Sejnowsky T. (eds.) *Proceedings of the 1988 Connectionist Models Summer School*, 21-28, Morgan Kaufmann (1988)

Le Cun Y., Boser B., & al., (1990): Handwritten Digit Recognition with a Back-Propagation Network- in D.Touretzky (ed.) *Advances in Neural Information Processing Systems 2* , 396-404, Morgan Kaufmann

Ljung L. & Söderström T. (1983) *Theory and Practice of Recursive Identification*, MIT Press

Tsypkin Ya. (1971) *Adaptation and Learning in Automatic systems*, Mathematics in science and engineering, vol 73, Academic Press

White H. (1991) An Overview of Representation and Convergence results for Multilayer feed-forward Networks, Touretzky D.S., Lipmann R. (eds.) *Advances in Neural Information Processing Systems 3* (this volume), Morgan-Kaufman
